# From Speech Recognition to Spoken Language Understanding: The Development of the MIT SUMMIT and VOYAGER Systems

Victor Zue, James Glass, David Goodine, Lynette Hirschman,
Hong Leung, Michael Phillips, Joseph Polifroni, and Stephanie Seneff
Room NE43-601
Spoken Language Systems Group
Laboratory for Computer Science
Massachusetts Institute of Technology
Cambridge, MA 02139 U.S.A.

## Abstract

Spoken language is one of the most natural, efficient, flexible, and economical means of communication among humans. As computers play an ever increasing role in our lives, it is important that we address the issue of providing a graceful human-machine interface through spoken language. In this paper, we will describe our recent efforts in moving beyond the scope of speech recognition into the realm of spoken-language understanding. Specifically, we report on the development of an urban navigation and exploration system called VOYAGER, an application which we have used as a basis for performing research in spoken-language understanding.

## 1  Introduction

Over the past decade, research in speech coding and synthesis has matured to the extent that speech can now be transmitted efficiently and generated with high intelligibility. Spoken input to computers, however, has yet to pass the threshold of practicality. Despite some recent successful demonstrations, current speech recognition systems typically fall far short of human capabilities of continuous speech recognition with essentially unrestricted vocabulary and speakers, under adverse acoustic environments. This is largely due to our incomplete knowledge of the encoding of linguistic information in the speech signal, and the inherent variabilities of

this process. Our approach to system development is to seek a good understanding of human communication through spoken language, to capture the essential features of the process in appropriate models, and to develop the necessary computational framework to make use of these models for machine understanding.

Our research in spoken language system development is based on the premise that many of the applications suitable for human/machine interaction using speech typically involve interactive problem solving. That is, in addition to converting the speech signal to text, the computer must also understand the user's request, in order to generate an appropriate response. As a result, we have focused our attention on three main issues. First, the system must operate in a realistic application domain, where domain-specific information can be utilized to translate spoken input into appropriate actions. The use of a realistic application is also critical to collecting data on how people would like to use machines to access information and solve problems. Use of a constrained task also makes possible rigorous evaluations of system performance. Second and perhaps most importantly the system must integrate speech recognition and natural language technologies to achieve speech *understanding*. Finally, the system must begin to deal with interactive speech, where the computer is an active conversational participant, and where people produce spontaneous speech, including false starts, hestitations, etc.

In this paper, we will describe our recent efforts in developing a spoken language interface for an urban navigation system (VOYAGER). We begin by describing our overall system architecture, paying particular attention to the interface between speech and natural language. We then describe the application domain and some of the issues that arise in realistic interactive problem solving applications, particulary in terms of conversational interaction. Finally, we report results of some performance evaluations we have made, using a spontaneous speech corpus we collected for this task.

## 2    System Architecture

Our spoken language language system contains three important components. The SUMMIT speech recognition system converts the speech signal into a set of word hypotheses. The TINA natural language system interacts with the speech recognizer in order to obtain a word string, as well as a linguistic interpretation of the utterance. A control strategy mediates between the recognizer and the language understanding component, using the language understanding constraints to help control the search of the speech recognition system.

### 2.1    Continuous Speech Recognition: The SUMMIT System

The SUMMIT system (Zue et al., 1989) starts the recognition process by first transforming the speech signal into a representation that models some of the known properties of the human auditory system (Seneff, 1988). Using the output of the auditory model, acoustic landmarks of varying robustness are located and embedded in a hierarchical structure called a dendrogram (Glass, 1988). The acoustic segments in the dendrogram are then mapped to phoneme hypotheses, using a set of automatically determined acoustic attributes in conjunction with conventional

pattern recognition algorithms. The result is a phoneme network, in which each arc is characterized by a vector of probabilities for all the possible candidates. Recently, we have begun to experiment with the use of artificial neural nets for phonetic classifiction. To date, we have been able to improve the system's classification performance by over 5% (Leung and Zue, 1990).

Words in the lexicon are represented as pronunciation networks, which are generated automatically by a set of phonological rules (Zue et al., 1990). Weights derived from training data are assigned to each arc, using a corrective training procedure, to reflect the likelihood of a particular pronunciation. Presently, lexical decoding is accomplished by using the Viterbi algorithm to find the best path that matches the acoustic-phonetic network with the lexical network.

## 2.2 Natural Language Processing: The TINA System

In a spoken language system, the natural language component should perform two critical functions: 1) provide constraint for the recognizer component, and 2) provide an interpretation of the meaning of the sentence to the back-end. Our natural language system, TINA, was specifically designed to meet these two needs. TINA is a probabilistic parser which operates top-down, using an agenda-based control strategy which favors the most likely analyses. The basic design of TINA has been described elsewhere (Seneff, 1989), but will be briefly reviewed. The grammar is entered as a set of simple context-free rules which are automatically converted to a shared network structure. The *nodes* in the network are augmented with constraint filters (both syntactic and semantic) that operate only on locally available parameters. All arcs in the network are associated with probabilities, acquired automatically from a set of training sentences. Note that the probabilities are established *not* on the rule productions but rather on arcs connecting sibling pairs in a shared structure for a number of linked rules. The effect of such pooling is essentially a hierarchical bigram model. We believe this mechanism offers the capability of generating probabilities in a reasonable way by sharing counts on syntactically/semantically identical units in differing structural environments.

## 2.3 Control Strategy

The current interface between the SUMMIT speech recognition system and the TINA natural language system, uses an $N$-best algorithm (Chow and Schwartz, 1989; Soong and Huang, 1990; Zue et al., 1990), in which the recognizer can propose its best $N$ complete sentence hypotheses one by one, stopping with the first sentence that is successfully analyzed by the natural language component TINA. In this case, TINA acts as a filter on *whole sentence* hypotheses.

In order to produce $N$-best hypotheses, we use a search strategy that involves an initial Viterbi search all the way to the end of the sentence, to provide a "best" hypothesis, followed by an $A^*$ search to produce next-best hypotheses in turn, provided that the first hypothesis failed to parse. If all hypotheses fail to parse the system produces the rejection message, "I'm sorry but I didn't understand you."

Even with the parser acting as a filter of whole-sentence hypotheses, it is appropriate to also provide the recognizer with an inexpensive language model that can partially

constrain the theories. This is currently done with a word-pair language model, in which each word in the vocabulary is associated with a list of words that could possibly follow that word *anywhere* in the sentence.

# 3    The VOYAGER Application Domain

VOYAGER is an urban navigation and exploration system that enables the user to ask about places of interest and obtain directions. It has been under development since early 1989 (Zue et al., 1989; Zue et al., 1990). In this section, we describe the application domain, the interface between our language understanding system TINA and the application back-end, and the discourse capabilities of the current system.

## 3.1    Domain Description

For our first attempt at exploring issues related to a fully-interactive spoken-language system, we selected a task in which the system knows about the physical environment of a specific geographical area and can provide assistance on how to get from one location to another within this area. The system, which we call VOYAGER, can also provide information concerning certain objects located inside this area. The current version of VOYAGER focuses on the geographic area of the city of Cambridge, MA between MIT and Harvard University.

The application database is an enhanced version of the Direction Assistance program developed at the Media Laboratory at MIT (Davis and Trobaugh, 1987). It consists of a map database, including the locations of various classes of objects (streets, buildings, rivers) and properties of these objects (address, phone number, etc.) The application supports a set of retrieval functions to access these data. The application must convert the semantic representation of TINA into the appropriate function call to the VOYAGER back-end. The answer is given to the user in three forms. It is graphically displayed on a map, with the object(s) of interest highlighted. In addition, a textual answer is printed on the screen, and is also spoken verbally using synthesized speech. The current implementation handles various types of queries, such as the location of objects, simple properties of objects, how to get from one place to another, and the distance and time for travel between objects.

## 3.2    Application Interface to VOYAGER

Once an utterance has been processed by the language understanding system, it is passed to an interface component which constructs a command function from the natural language representation. This function is subsequently passed to the back-end where a response is generated. There are three function types used in the current command framework of VOYAGER, which we will illustrate with the following example:

>    Query:    Where is the nearest bank to MIT?
> Function:    `(LOCATE (NEAREST (BANK nil) (SCHOOL "MIT")))`

LOCATE is an example of a major function that determines the primary action to be performed by the command. It shows the physical location of an object or set

of objects on the map. Functions such as BANK and SCHOOL in the above example access the database to return an object or a set of objects. When null arguments are provided, all possible candidates are returned from the database. Thus, for example, (SCHOOL "MIT") and (BANK nil) will return the objects MIT and all known banks, respectively. Finally, there are a number of functions in VOYAGER that act as filters, whereby the subset that fulfills some requirements are returned. The function (NEAREST X y), for example, returns the object in the set X that is closest to the object y. These filter functions can be nested, so that they can quite easily construct a complicated object. For example, "the Chinese restaurant on Main Street nearest to the hotel in Harvard Square that is closest to City Hall" would be represented by,

```
(NEAREST
  (ON-STREET
    (SERVE (RESTAURANT nil) "Chinese")
    (STREET "Main" "Street"))
  (NEAREST
    (IN-REGION (HOTEL nil) (SQUARE "Harvard"))
    (PUBLIC-BUILDING "City Hall")))
```

## 3.3  Discourse Capabilities

Carrying on a conversation requires the use of context and discourse history. Without context, some user input may appear underspecified, vague or even ill-formed. However, in context, these queries are generally easily understood. The discourse capabilities of the current VOYAGER system are simplistic but nonetheless effective in handling the majority of the interactions within the designated task. We describe briefly how a discourse history is maintained, and how the system keeps track of incomplete requests, querying the user for more information as needed to fill in ambiguous material.

Two slots are reserved for discourse history. The first slot refers to the location of the user, which can be set during the course of the conversation and then later referred to. The second slot refers to the most recently referenced set of objects. This slot can be a single object, a set of objects, or two separate objects in the case where the previous command involved a calculation involving both a source and a destination. With these slots, the system can process queries that include pronominal reference as in "What is their address?" or "How far is it from here?"

VOYAGER can also handle underspecified or vague queries, in which a function argument has either no value or multiple values. Examples of such queries would be "How far is a bank?" or "How far is MIT?" when no [FROM-LOCATION] has been specified. VOYAGER points out such underspecification to the user, by asking for specific clarification. The underspecified command is also pushed onto a stack of incompletely specified commands. When the user provides additional information that is evaluated successfully, the top command in the stack is popped for reevaluation. If the additional information is not sufficient to resolve the original command, the command is again pushed onto the stack, with the new information incorporated. A protection mechanism automatically clears the history stack whenever the user abandons a line of discussion before all underspecified queries are clarified.

## 4   Performance Evaluation

In this section, we describe our experience with performance evaluation of spoken language systems. The version of VOYAGER that we evaluated has a vocabulary of 350 words. The word-pair language model for the speech recognition sub-system has a perplexity of 72. For the *N*-best algorithm, the number of sentence hypotheses was arbitrarily set at 100. The system was implemented on a SUN-4, using four commercially available signal processing boards. This configuration has a processes an utterance in 3 to 5 times real-time.

The system was trained and tested using a corpus of spontaneous speech recorded from 50 male and 50 female subjects (Zue et al., 1989). We arbitrarily designated the data from 70 speakers, equally divided between male and female, to be the training set. Data from 20 of the remaining speakers were designated as the development set. The test set consisted of 485 utterances generated by the remaining 5 male and 5 female subjects. The average number of words per sentence was 7.7.

VOYAGER generated an action for 51.7% of the sentences in the test set. The system failed to generate a parse on the remaining 48.3% of the sentences, either due to recognizer errors, unknown words, unseen linguistic structures, or back-end inadequacy. Specifically, 20.3% failed to generate an action due to recognition errors or the system's inability to deal with spontaneous speech phenomena, 17.2% were found to contain unknown words, and an additional 10.5% would not have parsed even if recognized correctly. VOYAGER almost never failed to provide a response once a parse had been generated. This is a direct result of our conscious decision to constrain TINA according to the capabilities of the back-end. Although 48.3% of the sentences were judged to be incorrect, only 13% generated the wrong response. For the remainder of the errors, the system responded with the message, "I'm sorry but I didn't understand you."

Finally, we solicited judgments from three naive subjects who had had no previous experience with VOYAGER to assess the capabilities of the back-end. About 80% of the responses were judged to be appropriate, with an additional 5% being verbose but otherwise correct. Only about 4% of the sentences produced diagnostic error messages, for which the system was judged to give an appropriate response about two thirds of the time. The response was judged incorrect about 10% of the time. The subjects judged about 87% of the user queries to be reasonable.

## 5   Summary

This paper summarizes the status of our recent efforts in spoken language system development. It is clear that spoken language systems will incorporate research from, and provide a useful testbed for a variety of disciplines including speech, natural language processing, knowledge aquisition, databases, expert systems, and human factors. In the near term our plans include improving the phonetic recognition accuracy of SUMMIT by incorporating context-dependent models, and investigating control strategies which more fully integrate our speech recognition and natural language components.

## Acknowledgements

This research was supported by DARPA under Contract N00014-89-J-1332, monitored through the Office of Naval Research.

## References

Chow, Y, and R. Schwartz, (1989) "The N-Best Algorithm: An Efficient Procedure for Finding Top N Sentence Hypotheses", *Proc. DARPA Speech and Natural Language Workshop*, pp. 199-202, October.

Davis, J.R. and T. F. Trobaugh, (1987) "Back Seat Driver," Technical Report 1, MIT Media Laboratory Speech Group, December.

Glass, J. R., (1988) "Finding Acoustic Regularities in Speech: Applications to Phonetic Recognition," Ph.D. thesis, Massachusetts Institute of Technology, May.

Leung, H., and V. Zue, (1990) "Phonetic Classification Using Multi-Layer Perceptrons," *Proc. ICASSP-90*, pp. 525–528, Albuquerque, NM.

Seneff, S., (1988) "A Joint Synchrony/Mean-Rate Model of Auditory Speech Processing," *J. of Phonetics*, vol. 16, pp. 55–76, January.

Seneff, S. (1989) "TINA: A Probabilistic Syntactic Parser for Speech Understanding Systems," *Proc. DARPA Speech and Natural Language Workshop*, pp. 168–178, February.

Soong, F., and E. Huang, (1990) "A Tree-Trellis Based Fast Search for Finding the N-best Sentence Hypotheses in Continuous Speech Recognition", *Proc. DARPA Speech and Natural Language Workshop*, pp. 199-202, June.

Zue, V., J. Glass, M. Phillips, and S. Seneff, (1989) "Acoustic Segmentation and Phonetic Classification in the SUMMIT System," *Proc. ICASSP-89*, pp. 389–392, Glasgow, Scotland.

Zue, V., J. Glass, D. Goodine, H. Leung, M. Phillips, J. Polifroni, and S. Seneff, (1989) "The VOYAGER Speech Understanding System: A Progress Report," *Proc. DARPA Speech and Natural Language Workshop*, pp. 51–59, October.

Zue, V., N. Daly, J. Glass, D. Goodine, H. Leung, M. Phillips, J. Polifroni, S. Seneff, and M. Soclof, (1989) "The Collection and Preliminary Analysis of a Spontaneous Speech Database," *Proc. DARPA Speech and Natural Language Workshop*, pp. 126–134, October.

Zue, V., J. Glass, D. Goodine, M. Phillips, and S. Seneff, (1990) "The SUMMIT Speech Recognition System: Phonological Modelling and Lexical Access," *Proc. ICASSP-90*, pp. 49–52, Albuquerque, NM.

Zue, V., J. Glass, D. Goodine, H. Leung, M. Phillips, J. Polifroni, and S. Seneff, (1990) "The VOYAGER Speech Understanding System: Preliminary Development and Evaluation," *Proc. ICASSP-90*, pp. 73–76, Albuquerque, NM.

Zue, V., J. Glass, D. Goodine, H. Leung, M. Phillips, J. Polifroni, and S. Seneff, (1990) "Recent Progress on the VOYAGER System," *Proc. DARPA Speech and Natural Language Workshop*, pp. 206–211, June.